# Dynamic Infinite Relational Model for Time-varying Relational Data Analysis

**Katsuhiko Ishiguro**    **Tomoharu Iwata**    **Naonori Ueda**
NTT Communication Science Laboratories
Kyoto, 619-0237 Japan
{ishiguro,iwata,ueda}@cslab.kecl.ntt.co.jp

**Joshua Tenenbaum**
MIT
Boston, MA.
jbt@mit.edu

## Abstract

We propose a new probabilistic model for analyzing dynamic evolutions of relational data, such as additions, deletions and split & merge, of relation clusters like communities in social networks. Our proposed model abstracts observed time-varying object-object relationships into relationships between object clusters. We extend the infinite Hidden Markov model to follow dynamic and time-sensitive changes in the structure of the relational data and to estimate a number of clusters simultaneously. We show the usefulness of the model through experiments with synthetic and real-world data sets.

## 1   Introduction

Analysis of "relational data", such as the hyperlink structure on the Internet, friend links on social networks, or bibliographic citations between scientific articles, is useful in many aspects. Many statistical models for relational data have been presented [10, 1, 18]. The stochastic block model (SBM) [11] and the infinite relational model (IRM) [8] partition objects into clusters so that the relations between clusters abstract the relations between objects well. SBM requires specifying the number of clusters in advance, while IRM automatically estimates the number of clusters. Similarly, the mixed membership model [2] associates each object with multiple clusters (roles) rather than a single cluster.

These models treat the relations as *static* information. However, a large amount of relational data in the real world is *time-varying*. For example, hyperlinks on the Internet are not stationary since links disappear while new ones appear every day. Human relationships in a company sometimes drastically change by the splitting of an organization or the merging of some groups due to e.g. Mergers and Acquisitions. One of our modeling goals is to detect these sudden changes in network structure that occur over time.

Recently some researchers have investigated the dynamics in relational data. Tang et al.[13] proposed a spectral clustering-based model for multi-mode, time-evolving relations. Yang et al.[16] developed the time-varying SBM. They assumed a transition probability matrix like HMM, which governs all the cluster assignments of objects for all time steps. This model has only one transition probability matrix for the entire data. Thus, it cannot represent more complicated time variations such as split & merge of clusters that only occur temporarily. Fu et al.[4] proposed a time-series extension of the mixed membership model. [4] assumes a continuous world view: roles follow a mixed membership structure; model parameters evolve continuously in time. This model is very general for time series relational data modeling, and is good for tracking gradual and continuous changes of the relationships. Some works in bioinformatics [17, 5] have also adopted similar strategies. However, a continuous model approach does not necessarily best capture sudden transitions of the relationships we are interested in. In addition, previous models assume the number of clusters is fixed and known, which is difficult to determine a priori.

In this paper we propose yet another time-varying relational data model that deals with temporal and dynamic changes of cluster structures such as additions, deletions and split & merge of clusters. Instead of the continuous world view of [4], we assume a discrete structure: distinct clusters with discrete transitions over time, allowing for birth, death and split & merge dynamics. More specifically, we extend IRM for time-varying relational data by using a variant of the infinite HMM (iHMM) [15, 3]. By incorporating the idea of iHMM, our model is able to infer clusters of objects without specifying a number of clusters in advance. Furthermore, we assume multiple transition probabilities that are dependent on time steps and clusters. This specific form of iHMM enables the model to represent time-sensitive dynamic properties such as split & merge of clusters. Inference is performed efficiently with the slice sampler.

## 2 Infinite Relational Model

We first explain the infinite relational model (IRM) [8], which can estimate the number of hidden clusters from a relational data. In IRM, Dirichlet process (DP) is used as a prior for clusters of an unknown number, and is denoted as $DP(\gamma, G_0)$ where $\gamma > 0$ is a parameter and $G_0$ is a base measure. We write $G \sim DP(\gamma, G_0)$ when a distribution $G(\theta)$ is sampled from DP. In this paper, we implement DP by using a stick-breaking process [12], which is based on the fact that $G$ is represented as an infinite mixture of $\theta$s: $G(\theta) = \sum_{k=1}^{\infty} \beta_k \delta_{\theta_k}(\theta), \theta_k \sim G_0$. $\boldsymbol{\beta} = (\beta_1, \beta_2, \ldots)$ is a mixing ratio vector with infinite elements whose sum equals one, constructed in a stochastic way:

$$\beta_k = v_k \prod_{l=1}^{k-1} (1 - v_l), \quad v_k \sim \text{Beta}(1, \gamma). \tag{1}$$

Here $v_k$ is drawn from a Beta distribution with a parameter $\gamma$.

The IRM is an application of the DP for relational data. Let us assume a binary two-place relation on the set of objects $D = \{1, 2, \ldots, N\}$ as $D \times D \to \{0, 1\}$. For simplicity, we only discuss a two place relation between the identical domain ($D \times D$). The IRM divides the set of $N$ objects into multiple clusters based on the observed relational data $X = \{x_{i,j} \in \{0, 1\}; \ 1 \le i, j \le N\}$. The IRM is able to infer the number of clusters at the same time because it uses DP as a prior distribution of the cluster partition. Observation $x_{i,j} \in \{0, 1\}$ denotes the existence of a relation between objects $i, j \in \{1, 2, \ldots, N\}$. If there is (not) a relation between $i$ and $j$, then $x_{i,j} = 1$ (0). We allow asymmetric relations $x_{i,j} \neq x_{j,i}$ throughout the paper.

The probabilistic generative model (Fig. 1(a)) of the IRM is as follows:

$$\boldsymbol{\beta}|\gamma \sim \text{Stick}(\gamma) \tag{2}$$
$$z_i|\boldsymbol{\beta} \sim \text{Multinomial}(\boldsymbol{\beta}) \tag{3}$$
$$\eta_{k,l}|\xi, \psi \sim \text{Beta}(\xi, \psi) \tag{4}$$
$$x_{i,j}|Z, H \sim \text{Bernoulli}\left(\eta_{z_i, z_j}\right). \tag{5}$$

Here, $Z = \{z_i\}_{i=1}^{N}$ and $H = \{\eta_{k,l}\}_{k,l=1}^{\infty}$. In Eq. (2) "Stick" is the stick-breaking process (Eq. (1)). We sample a cluster index of the object $i$, $z_i = k, k \in \{1, 2, \ldots, \}$ using $\boldsymbol{\beta}$ as in Eq. (3). In Eq. (4) $\eta_{k,l}$ is the strength of a relation between the objects in clusters $k$ and $l$. Generating the observed relational data $x_{i,j}$ follows Eq. (5) conditioned by the cluster assignments $Z$ and the strengths $H$.

## 3 Dynamic Infinite Relational Model (dIRM)

### 3.1 Time-varying relational data

First, we define the time-varying relational data considered in this paper. Time-varying relational data $X$ have three subscripts $t$, $i$, and $j$: $X = \left\{x_{t,i,j} \in \{0, 1\}\right\}$, where $i, j \in \{1, 2, \ldots, N\}$, $t \in \{1, 2, \ldots, T\}$. $x_{t,i,j} = 1(0)$ indicates that there is (not) an observed relationship between objects $i$ and $j$ at time step $t$. $T$ is the number of time steps, and $N$ is the number of objects. We assume that there is no relation between objects belonging to a different time step $t$ and $t'$. The time-varying relational data $X$ is a set of $T$ (static) relational data for $T$ time steps.

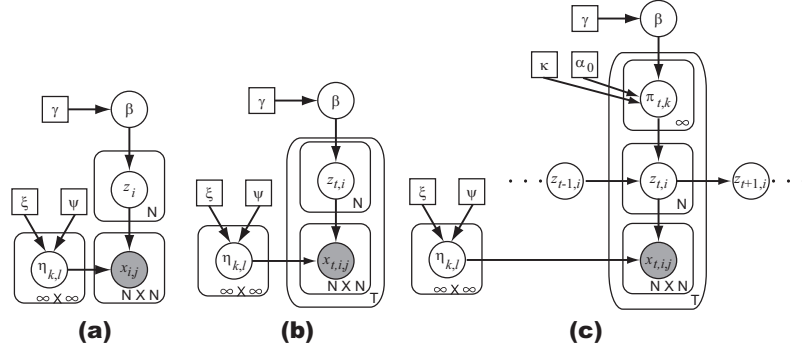

Figure 1: Graphical model of (a)IRM (Eqs.2-5), (b)"tIRM" (Eqs.7-10), and (c)dIRM (Eqs.11-15). Circle nodes denote variables, square nodes are constants and shaded nodes indicate observations.

It is natural to assume that every object transits between different clusters along with the time evolution. Observing several real world time-varying relational data, we assume there are several properties of transitions, as follows:

- P1. Cluster assignments in consecutive time steps have higher correlations.

- P2. Time evolutions of clusters are not stationary nor uniform.

- P3. The number of clusters is time-varying and unknown a priori.

P1 is a common assumption for many kinds of time series data, not limited to relational data. For example, a member of a firm community on SNSs will belong to the same community for a long time. A hyperlink structure in a news website may alter because of breaking news, but most of the site does not change as rapidly every minute.

P2 tries to model occasional and drastic changes from frequent and minor modifications in relational networks. Such unstable changes are observed elsewhere. For example, human relationships in companies will evolve every day, but a merger of departments sometimes brings about drastic changes. On an SNS, a user community for the upcoming Olympics games may exist for a limited time: it will not last years after the games end. This will cause an addition and deletion of a user cluster (community). P3 is indispensable to track such changes of clusters.

## 3.2 Naive extensions of IRM

We attempt to modify the IRM to satisfy these properties. We first consider several straightforward solutions based on the IRM for analyzing time-varying relational data.

The simplest way is to convert time-varying relational data $X$ into "static" relational data $\tilde{X} = \{\tilde{x}_{i,j}\}$ and apply the IRM to $\tilde{X}$. For example, we can generate $\tilde{X}$ as follows:

$$\tilde{x}_{i,j} = \begin{cases} 1 & \frac{1}{T}\sum_{t=1}^{T} x_{t,i,j} > \sigma, \\ 0 & \text{otherwise}, \end{cases} \qquad (6)$$

where $\sigma$ denotes a threshold. This solution cannot represent the time changes of clustering because it assume the same clustering results for all the time steps ($z_{1,i} = z_{2,i} = \cdots = z_{T,i}$).

We may separate the time-varying relational data $X$ into a series of time step-wise relational data $X_t$ and apply the IRM for each $X_t$. In this case, we will have a different clustering result for each time step, but the analysis ignores the dependency of the data over time.

Another solution is to extend the object assignment variable $z_i$ to be time-dependent $z_{t,i}$. The resulting "tIRM" model is described as follows (Fig. 1(b)):

$$\boldsymbol{\beta}|\gamma \sim \text{Stick}(\gamma) \tag{7}$$

$$z_{t,i}|\boldsymbol{\beta} \sim \text{Multinomial}(\boldsymbol{\beta}) \tag{8}$$

$$\eta_{k,l}|\xi,\psi \sim \text{Beta}(\xi,\psi) \tag{9}$$

$$x_{t,i,j}|Z_t, H \sim \text{Bernoulli}\left(\eta_{z_{t,i},z_{t,j}}\right). \tag{10}$$

Here, $Z_t = \{z_{t,i}\}_{i=1}^N$. Since $\boldsymbol{\beta}$ is shared over all time steps, we may expect that the clustering results between time steps will have higher correlations. However, this model assumes that $z_{t,i}$ is conditionally independent from each other for all $t$ given $\boldsymbol{\beta}$. This implies that the tIRM is not suitable for modeling time evolutions since the order of time steps are ignored in the model.

## 3.3   dynamic IRM

To address three conditions P1∼3 above, we propose a new probabilistic model called the dynamic infinite relational model (dIRM). The generative model is given below:

$$\boldsymbol{\beta}|\gamma \sim \text{Stick}(\gamma) \tag{11}$$

$$\boldsymbol{\pi}_{t,k}|\alpha_0, \kappa, \boldsymbol{\beta} \sim \text{DP}\left(\alpha_0 + \kappa, \frac{\alpha_0\boldsymbol{\beta} + \kappa\boldsymbol{\delta}_k}{\alpha_0 + \kappa}\right) \tag{12}$$

$$z_{t,i}|z_{t-1,i}, \Pi_t \sim \text{Multinomial}\left(\boldsymbol{\pi}_{t,z_{t-1,i}}\right) \tag{13}$$

$$\eta_{k,l}|\xi,\psi \sim \text{Beta}(\xi,\psi) \tag{14}$$

$$x_{t,i,j}|Z_t, H \sim \text{Bernoulli}\left(\eta_{z_{t,i},z_{t,j}}\right). \tag{15}$$

Here, $\Pi_t = \{\boldsymbol{\pi}_{t,k} : k = 1, \ldots, \infty\}$. A graphical model of the dIRM is presented in Fig. 1(c).

$\boldsymbol{\beta}$ in Eq. (11) represents time-average memberships (mixing ratios) to clusters. Newly introduced $\boldsymbol{\pi}_{t,k} = (\pi_{t,k,1}, \pi_{t,k,2}, \ldots, \pi_{t,k,l}, \ldots)$ in Eq. (12) is a transition probability that an object remaining in the cluster $k \in \{1, 2, \ldots\}$ at time $t-1$ will move to the cluster $l \in \{1, 2, \ldots\}$ at time $t$. Because of the DP, this transition probability is able to handle infinite hidden states like iHMM [14].

The DP used in Eq. (12) has an additional term $\kappa > 0$, which is introduced by Fox et al. [3]. $\boldsymbol{\delta}_k$ is a vector whose elements are zero except the $k$th element, which is one. Because the base measure in Eq. (12) is biased by $\kappa$ and $\boldsymbol{\delta}_k$, the $k$th element of $\boldsymbol{\pi}_{t,k}$ prefers to take a larger value than other elements. This implies that this DP encourages the self-transitions of objects, and we can achieve the property P1 for time-varying relational data.

One difference from conventional iHMMs [14, 3] lies in P2, which is achieved by making the transition probability $\boldsymbol{\pi}$ time-dependent. $\boldsymbol{\pi}_{t,k}$ is sampled for every time step $t$, thus, we can model time-varying patterns of transitions, including additions, deletions and split & merge of clusters as extreme cases. These changes happen only temporarily, therefore, time-dependent transition probabilities are indispensable for our purpose. Note that the transition probability is also dependent on the cluster index $k$, as in conventional iHMMs. Also the dIRM can automatically determine the number of clusters thanks to DP: this enables us to hold P3.

Equation (13) generates a cluster assignment for the object $i$ at time $t$, based on the cluster, where the object was previously ($z_{t-1,i}$) and its transition probability $\boldsymbol{\pi}$. Equation (14) generates a strength parameter $\eta$ for the pair of clusters $k$ and $l$, then we obtain the observed sample $x_{t,i,j}$ in Eq. (15).

The difference between iHMMs and dIRM is two-fold. One is the time-dependent transition probability of the dIRM discussed above. The another is that the iHMMs have one hidden state sequence $s_{1:t}$ to be inferred, while the dIRM needs to estimate multiple hidden state sequences $z_{1:t,i}$ given one time sequence observation. Thus, we may interpret the dIRM as an extension of the iHMM, which has $N$ (= a number of objects) hidden sequences to handle relational data.

# 4 Inference

We use a slice sampler [15], which enables fast and efficient sampling of the sequential hidden states. The slice sampler introduces auxiliary variables $U = \{u_{t,i}\}$. Given $U$, the number of clusters can be reduced to a finite number during the inference, and it enables us an efficient sampling of variables.

## 4.1 Sampling parameters

First, we explain the sampling of an auxiliary variable $u_{t,i}$. We assume a prior of $u_{t,i}$ as a uniform distribution. Also we define the joint distribution of $u$, $z$, and $x$:

$$p\left(x_{t,i,j}, u_{t,i}, u_{t,j}, z_{t-1:t,i}, z_{t-1:t,j}\right) = \mathbb{I}\left(u_{t,i} < \pi_{t,z_{t-1,i},z_{t,i}}\right) \mathbb{I}\left(u_{t,j} < \pi_{t,z_{t-1,j},z_{t,j}}\right) x_{t,i,j}^{\eta_{z_{t,i},z_{t,j}}} \left(1 - x_{t,i,j}\right)^{1-\eta_{z_{t,i},z_{t,j}}} . \quad (16)$$

Here, $\mathbb{I}(\cdot)$ is 1 if the predicate holds, otherwise zero. Using Eq. (16), we can derive the posterior of $u_{t,i}$ as follows:

$$u_{t,i} \sim \text{Uniform}\left(0, \pi_{t,z_{t-1,i},z_{t,i}}\right). \quad (17)$$

Next, we explain the sampling of an object assignment variable $z_{t,i}$. We define the following message variable $p$:

$$p_{t,i,k} = p\left(z_{t,i} = k | X_{1:t}, U_{1:t}, \Pi, H, \boldsymbol{\beta}\right). \quad (18)$$

Sampling of $z_{t,i}$ is similar to the forward-backward algorithm for the original HMM. First, we compute the above message variables from $t = 1$ to $t = T$ (forward filtering). Next, we sample $z_{t,i}$ from $t = T$ to $t = 1$ using the computed message variables (backward sampling).

In forward filtering we compute the following equation from $t = 1$ to $t = T$:

$$p_{t,i,k} \propto p\left(x_{t,i,i} | z_{t,i} = k, H\right) \prod_{j \neq i} p\left(x_{t,i,j} | z_{t,i} = k, H\right) p\left(x_{t,j,i} | z_{t,i} = k, H\right) \sum_{l:u_{t,i} < \pi_{t,l,k}} p_{t-1,i,l}. \quad (19)$$

Note that the summation is conditioned by $u_{t,i}$. The number of $l$s (cluster indices) that hold this condition is limited to a certain finite number. Thus, we can evaluate the above equation.

In backward sampling, we sample $z_{t,i}$ from $t = T$ to $t = 1$ from the equation below:

$$p\left(z_{t,i} = k | z_{t+1,i} = l\right) \propto p_{t,i,k} \pi_{t+1,k,l} \mathbb{I}\left(u_{t+1,i} < \pi_{t+1,k,l}\right). \quad (20)$$

Because of $\mathbb{I}(u < \pi)$, values of cluster indices $k$ are limited within a finite set. Therefore, the variety of sampled $z_{t,i}$ will be limited a certain finite number $K$ given $U$.

Given $U$ and $Z$, we have finite $K$-realized clusters. Thus, computing the posteriors of $\boldsymbol{\pi}_{t,k}$ and $\eta_{k,l}$ becomes easy and straightforward. First $\boldsymbol{\beta}$ is assumed as a $K + 1$-dimensional vector (mixing ratios of unrepresented clusters are aggregated in $\beta_{K+1} = 1 - \sum_{k=1}^{K} \beta_k$). $m_{t,k,l}$ denotes a number of objects $i$ such that $z_{t-1,i} = k$ and $z_{t,i} = l$. Also, let us denote a number of $x_{t,i,j}$ such that $z_{t,i} = k$ and $z_{t,j} = l$ as $N_{k,l}$. Similarly, $n_{k,l}$ denotes a number of $x_{t,i,j}$ such that $z_{t,i} = k$, $z_{t,j} = l$ and $x_{t,i,j} = 1$. Then we obtain following posteriors:

$$\boldsymbol{\pi}_{t,k} \sim \text{Dirichlet}\left(\alpha_0 \boldsymbol{\beta} + \kappa \boldsymbol{\delta}_k + \boldsymbol{m}_{t,k}\right). \quad (21)$$

$$\eta_{k,l} \sim \text{Beta}\left(\xi + n_{k,l}, \psi + N_{k,l} - n_{k,l}\right). \quad (22)$$

$\boldsymbol{m}_{t,k}$ is a $K + 1$-dimensional vector whose $l$th element is $m_{t,k,l}$ ($m_{t,k,K+1} = 0$).

We omit the derivation of the posterior of $\boldsymbol{\beta}$ since it is almost the same with that of Fox et al. [3].

## 4.2 Sampling hyperparameters

Sampling hyperparameters is important to obtain the best results. This could be done normally by putting vague prior distributions [14]. However, it is difficult to evaluate the precise posteriors for some hyperparameters [3]. Instead, we reparameterize and sample a hyperparameter in terms of $a \in (0, 1)$ [6]. For example, if the hyperparameter $\gamma$ is assumed as Gamma-distributed, we convert $\gamma$ by $a = \frac{\gamma}{1+\gamma}$. Sampling $a$ can be achieved from a uniform grid on $(0, 1)$. We compute (unnormalized) posterior probability densities at several $a$s and choose one to update the hyperparameter.

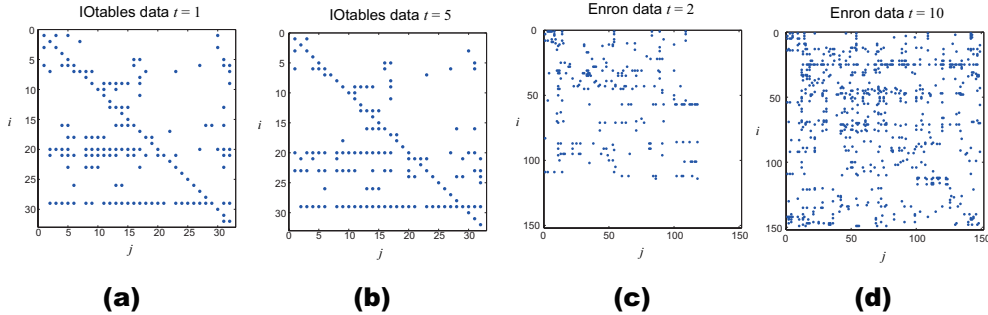

Figure 2: Example of real-world datasets. (a)IOtables data, observations at $t = 1$, (b)IOtables data, observations at $t = 5$, (c)Enron data, observations at $t = 2$, and (d)Enron data, observations at $t = 10$.

## 5  Experiments

Performance of the dIRM is compared with the original IRM [8] and its naive extension tIRM (described in Eqs. (7-10)). To apply the IRM to time-varying relational data, we use Eq. (6) to $X$ with a threshold $\sigma = 0.5$. The difference between the tIRM (Eqs. (7-10)) and the dIRM is that the tIRM does not incorporate the dependency between successive time steps while the dIRM does. Hyperparameters were estimated simultaneously in all experiments.

### 5.1  Datasets and measurements

We prepared two synthetic datasets (Synth1 and Synth2). To synthesize datasets, we first determined the number of time steps $T$, the number of clusters $K$, and the number of objects $N$. Next, we manually assigned $z_{t,i}$ in order to obtain cluster split & merge, additions, and deletions. After obtaining $Z$, we defined the connection strengths between clusters $H = \{\eta_{k,l}\}$. In this experiment, each $\eta_{k,l}$ may take one of two values $\eta = 0.1$ (weakly connected) or $\eta = 0.9$ (strongly connected). Observation $X$ was randomly generated according to $Z$ and $H$. Synth1 is smaller ($N = 16$) and stable while Synth2 is much larger ($N = 54$), and objects actively transit between clusters.

Two real-world datasets were also collected. The first one is the National Input-Output Tables for Japan (IOtables) provided by the Statistics Bureau of the Ministry of Internal Affairs and Communications of Japan. IOtables summarize the transactions of goods and services between industrial sectors. We used an inverted coefficient matrix, which is a part of the IOtables. Each element in the matrix $e_{i,j}$ represents that one unit of demand in the $j$th sector invokes $e_{i,j}$ productions in the $i$th sector. We generated $x_{i,j}$ from $e_{i,j}$ by binarizaion: setting $x_{i,j} = 1$ if $e_{i,j}$ exceeds the average, and setting $x_{i,j} = 0$ otherwise. We collected data from 1985, 1990, 1995, 2000, and 2005, in 32 sectors resolutions. Thus we obtain a time-varying relational data of $N = 32$ and $T = 5$.

The another real-world dataset is the Enron e-mail dataset [9], used in many studies including [13, 4]. We extracted e-mails sent in 2001. The number of time steps was $T = 12$, so the dataset was divided into monthly transactions. The full dataset contained $N = 151$ persons. $x_{t,i,j} = 1(0)$ if there is (not) an e-mail sent from $i$ to $j$ at time (month) $t$. We also generated a smaller dataset ($N = 68$) by excluding those who send few e-mails for convenience. Quantitative measurements were computed with this smaller dataset.

Fig. 2 presents examples of IOtables dataset ((a),(b)) and Enron dataset ((c),(d)). IOtables dataset characterized by its stable relationships, compared to Enron dataset. In Enron dataset, the amount of communication rapidly increases after the media reported on the Enron scandals.

We used three evaluating measurements. One is the Rand index, which computes the similarity between true and estimated clustering results [7]. The Rand index takes the maximum value (1) if the two clustering results completely match. We computed the Rand index between the ground truth $Z_t$ and the estimated $\hat{Z}_t$ for each time step, and averaged the indices for $T$ steps. We also compute the error in the number of estimated clusters. Differences in the number of realized clusters were computed between $Z_t$ and $\hat{Z}_t$, and we calculated the average of these errors for $T$ steps. We

Table 1: Computed Rand indices, numbers of erroneous clusters, and averaged test data log likelihoods.

| Data | Rand index | | | # of erroneous clusters | | | Test log likelihood | | |
|---|---|---|---|---|---|---|---|---|---|
| | IRM | tIRM | dIRM | IRM | tIRM | dIRM | IRM | tIRM | dIRM |
| Synth1 | 0.796 | 0.946 | **0.982** | 1.00 | 0.20 | **0.13** | -0.542 | -0.508 | **-0.505** |
| Synth2 | 0.433 | 0.734 | **0.847** | 3.00 | 0.98 | **0.65** | -0.692 | -0.393 | **-0.318** |
| IOtables | - | - | - | - | - | - | -0.354 | -0.358 | **-0.291** |
| Enron | - | - | - | - | - | - | -0.120 | -0.135 | **-0.106** |

calculated these measurements for the synthetic datasets. The third measure is an (approximated) test-data log likelihood. For all datasets, we generated noisy datasets whose observation values are inverted. The number of inverted elements was kept small so that inversions would not affect the global clustering results. The ratios of inverted elements over the entire elements are set to 5% for two synthetic data, 1% for IOtables data and 0.5% for Enron data. We made inferences on the noisy datasets, and computed the likelihoods that "inverted observations take the real value". We used the averaged log-likelihood per a observation as a measurement.

## 5.2 Results

First, we present the quantitative results. Table 1 lists the computed Rand index, errors in the estimated number of clusters, and test-data log likelihoods. We confirmed that dIRM outperformed the other models in all datasets for the all measures. Particularly, dIRM showed good results in the Synth2 and Enron datasets, where the changes in relationships are highly dynamic and unstable. On the other hand, the dIRM did not achieve a remarkable improvement against tIRM for the Synth1 dataset whose temporal changes are small. Thus we can say that the dIRM is superior in modeling time-varying relational data, especially for dynamic ones.

Next, we evaluate results of the real-world datasets qualitatively. Figure 3 shows the results from IOtables data. The panel (a) illustrates the estimated $\eta_{k,l}$ using the dIRM, and the panel (b) presents the time evolution of cluster assignments, respectively. The dIRM obtained some reasonable and stable industrial clusters, as shown in Fig. 3 (b). For example, dIRM groups the machine industries into cluster 5, and infrastructure related industries are grouped into cluster 13. We believe that the self-transition bias $\kappa$ helps the model find these stable clusters. Also relationships between clusters presented in Fig. 3 (a) are intuitively understandable. For example, demands for machine industries (cluster 5) will cause large productions for "iron and steel" sector (cluster 7). The "commerce & trade" and "enterprise services" sectors (cluster 10) connects strongly to almost all the sectors.

There are some interesting cluster transitions. First, look at the "finance, insurance" sector. At $t = 1$, this sector belongs to cluster 14. However, the sector transits to cluster 1 afterwards, which does not connect strongly with clusters 5 and 7. This may indicates the shift of money from these matured industries. Next, the "transport" sector enlarges its roll in the market by moving to cluster 14, and it causes the deletion of cluster 8. Finally, note the transitions of "telecom, broadcast" sector. From 1985 to 2000, this sector is in the cluster 9 which is rather independent from other clusters. However, in 2005 the cluster separated, and telecom industry merged with cluster 1, which is a influential cluster. This result is consistent with the rapid growth in ITC technologies and its large impact on the world.

Finally, we discuss results on the Enron dataset. Because this e-mail dataset contains many individuals' names, we refrain from cataloging the object assignments as in the IOtables dataset. Figure 4 (a) tells us that clusters 1 ~ 7 are relatively separated communities. For example, members in cluster 4 belong to a restricted domain business such energy, gas, or pipeline businesses. Cluster 5 is a community of financial and monetary departments, and cluster 7 is a community of managers such as vice presidents, and CFOs.

One interesting result from the dIRM is finding cluster 9. This cluster notably sends many messages to other clusters, especially for management cluster 7. The number of objects belonging to this cluster is only three throughout the time steps, but these members are the key-persons at that time.

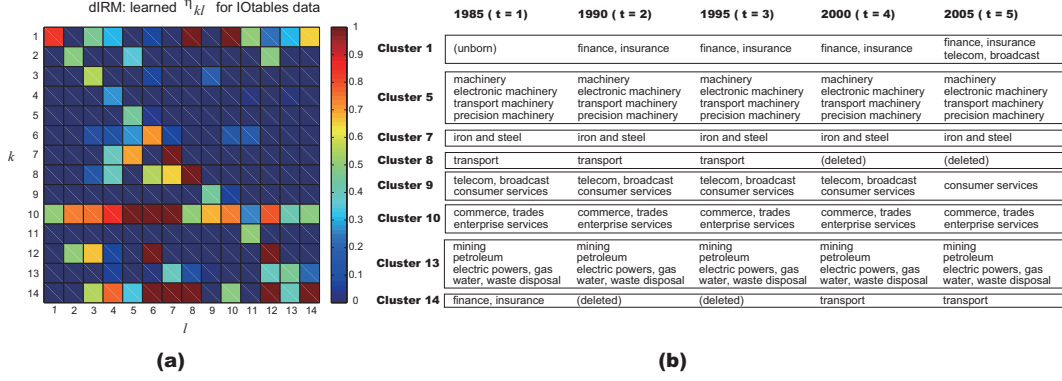

**(a)**                                            **(b)**

Figure 3: (a) Example of estimated $\eta_{k,l}$ (strength of relationship between clusters $k, l$) for IOtable data by dIRM. (d) Time-varying clustering assignments for selected clusters by dIRM.

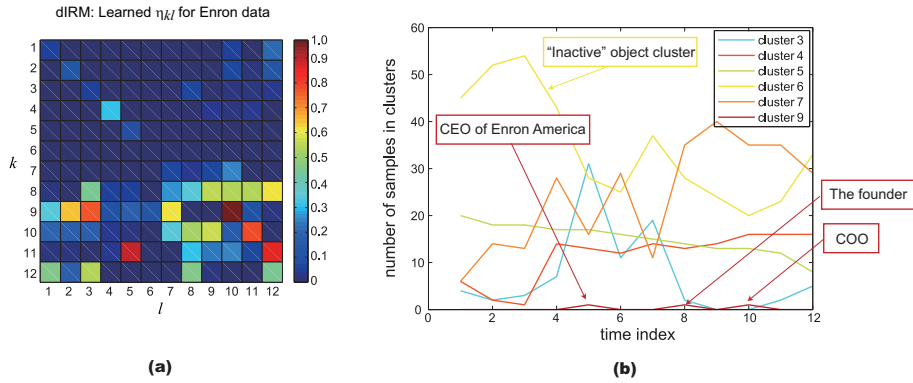

**(a)**                                            **(b)**

Figure 4: (a): Example of estimated $\eta_{k,l}$ for Enron dataset using dIRM. (b): Number of items belonging to clusters at each time step for Enron dataset using dIRM.

First, the CEO of Enron America stayed at cluster 9 in May ($t = 5$). Next, the founder of Enron was a member of the cluster in August $t = 8$. The CEO of Enron resigned that month, and the founder actually made an announcement to calm down the public. Finally, the COO belongs to the cluster in October $t = 10$. This is the month that newspapers reported the accounting violations.

Fig. 4 (b) presents the time evolutions of the cluster memberships; i.e. the number of objects belonging to each cluster at each time step. In contrast to the IOtables dataset, this Enron e-mail dataset is very dynamic, as you can see from Fig. 2(c), (d). For example, the volume of cluster 6 (inactive cluster) decreases as time evolves. This result reflects the fact that the transactions between employees increase as the scandal is more and more revealed. On the contrary, cluster 4 is stable in membership. Thus, we can imagine that the group of energy and gas is a dense and strong community. This is also true for cluster 5.

## 6   Conclusions

We proposed a new time-varying relational data model that is able to represent dynamic changes of cluster structures. The dynamic IRM (dIRM) model incorporates a variant of the iHMM model and represents time-sensitive dynamic properties such as split & merge of clusters. We explained a generative model of the dIRM, and showed an inference algorithm based on a slice sampler. Experiments with synthetic and real-world time series datasets showed that the proposed model improves the precision of time-varying relational data analysis. We will apply this model to other datasets to study the capability and the reliability of the model. We also are interested in modifying the dIRM to deal with multi-valued observation data.

# References

[1] A. Clauset, C. Moore, and M. E. J. Newman. Hierarchical structure and the prediction of missing links in networks. *Nature*, 453:98–101, 2008.

[2] E. Erosheva, S. Fienberg, and J. Lafferty. Mixed-membership models of scientific publications. *Proceedings of the National Academy of Sciences of the United States of America (PNAS)*, 101(Suppl 1):5220–5227, 2004.

[3] E.B. Fox, E.B. Sudderth, M.I. Jordan, and A.S. Willsky. An HDP-HMM for systems with state persistence. In *Proceedings of the 25th International Conference on Machine Learning (ICML)*, 2008.

[4] Wenjie Fu, Le Song, and Eric P. Xing. Dynamic mixed membership blockmodel for evolving networks. In *Proceedings of the 26th International Conference on Machine Learning (ICML)*, 2009.

[5] O. Hirose, R. Yoshida, S. Imoto, R. Yamaguchi, T. Higuchi, D. S. Chamock-Jones, C. Print, and S. Miyano. Statistical inference of transcriptional module-based gene networks from time course gene expression profiles by using state space models. *Bioinformatics*, 24(7):932–942, 2008.

[6] P. D. Hoff. Subset clustering of binary sequences, with an application to genomic abnormality data. *Biometrics*, 61(4):1027–1036, 2005.

[7] L. Hubert and P. Arabie. Comparing partitions. *Journal of Classification*, 2(1):193–218, 1985.

[8] C. Kemp, J. B. Tenenbaum, T. L. Griffiths, T. Yamada, and N. Ueda. Learning systems of concepts with an infinite relational model. In *Proceedings of the 21st National Conference on Artificial Intelligence (AAAI)*, 2006.

[9] B. Klimat and Y. Yang. The enron corpus: A new dataset for email classification research. In *Proceedings of the European Conference on Machine Learning (ECML)*, 2004.

[10] D. Liben-Nowell and J. Kleinberg. The link prediction problem for social networks. In *Proceedings of the Twelfth International Conference on Information and Knowledge Management*, pages 556–559. ACM, 2003.

[11] K. Nowicki and T. A. B. Snijders. Estimation and prediction for stochastic blockstructures. *Journal of the American Statistical Association*, 96(455):1077–1087, 2001.

[12] J. Sethuraman. A constructive definition of dirichlet process. *Statistica Sinica*, 4:639–650, 1994.

[13] L. Tang, H. Liu, J. Zhang, and Z. Nazeri. Community evolution in dynamic multi-mode networks. In *Proceeding of the 14th ACM SIGKDD International Conference on Knowledge Discovery and Data Mining*, pages 677–685, 2008.

[14] Y. W. Teh, M. I. Jordan, M. J. Beal, and D. M. Blei. Hierarchical Dirichlet process. *Journal of The American Statistical Association*, 101(476):1566–1581, 2006.

[15] J. Van Gael, Y. Saatci, Y. W. Teh, and Z. Ghahramani. Beam sampling for the infinite hidden Markov model. In *Proceedings of the 25th International Conference on Machine Learning (ICML)*, 2008.

[16] T. Yang, Y. Chi, S. Zhu, Y. Gong, and R. Jin. A Bayesian approach toward finding communities and their evolutions in dynamic social networks. In *Proceedings of SIAM International Conference on Data Mining (SDM)*, 2009.

[17] R. Yoshida, S. Imoto, and T. Higuchi. Estimating time-dependent gene networks from time series microarray data by dynamic linear models with markov switching. In *Proceedings of the International Conference on Computational Systems Bioinformatics*, 2005.

[18] S. Zhu, K. Yu, and Y. Gong. Stochastic relational models for large-scale dyadic data using mcmc. In *Advances in Neural Information Processing Systems 21 (NIPS)*, 2009.

